# A general agnostic active learning algorithm

**Sanjoy Dasgupta**
UC San Diego
dasgupta@cs.ucsd.edu

**Daniel Hsu**
UC San Diego
djhsu@cs.ucsd.edu

**Claire Monteleoni**
UC San Diego
cmontel@cs.ucsd.edu

## Abstract

We present an agnostic active learning algorithm for any hypothesis class of bounded VC dimension under arbitrary data distributions. Most previous work on active learning either makes strong distributional assumptions, or else is computationally prohibitive. Our algorithm extends the simple scheme of Cohn, Atlas, and Ladner [1] to the agnostic setting, using reductions to supervised learning that harness generalization bounds in a simple but subtle manner. We provide a fall-back guarantee that bounds the algorithm's label complexity by the agnostic PAC sample complexity. Our analysis yields asymptotic label complexity improvements for certain hypothesis classes and distributions. We also demonstrate improvements experimentally.

## 1 Introduction

*Active learning* addresses the issue that, in many applications, labeled data typically comes at a higher cost (e.g. in time, effort) than unlabeled data. An active learner is given unlabeled data and must pay to view any label. The hope is that significantly fewer labeled examples are used than in the supervised (non-active) learning model. Active learning applies to a range of data-rich problems such as genomic sequence annotation and speech recognition. In this paper we formalize, extend, and provide label complexity guarantees for one of the earliest and simplest approaches to active learning—one due to Cohn, Atlas, and Ladner [1].

The scheme of [1] examines data one by one in a stream and requests the label of any data point about which it is currently unsure. For example, suppose the hypothesis class consists of linear separators in the plane, and assume that the data is linearly separable. Let the first six data be labeled as follows.

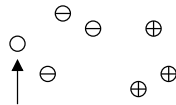

The learner does not need to request the label of the seventh point (indicated by the arrow) because it is not unsure about the label: any straight line with the $\oplus$s and $\ominus$s on opposite sides has the seventh point with the $\ominus$s. Put another way, the point is not in the *region of uncertainty* [1], the portion of the data space for which there is disagreement among hypotheses consistent with the present labeled data.

Although very elegant and intuitive, this approach to active learning faces two problems:

1. Explicitly maintaining the region of uncertainty can be computationally cumbersome.

2. Data is usually not perfectly separable.

Our main contribution is to address these problems. We provide a simple generalization of the selective sampling scheme of [1] that tolerates adversarial noise and never requests many more labels than a standard agnostic supervised learner would to learn a hypothesis with the same error.

In the previous example, an *agnostic* active learner (one that does not assume a perfect separator exists) is actually *still* uncertain about the label of the seventh point, because all six of the previous labels could be inconsistent with the best separator. Therefore, it should still request the label. On the other hand, after enough points have been labeled, if an unlabeled point occurs at the position shown below, chances are its label is not needed.

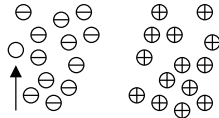

To extend the notion of uncertainty to the agnostic setting, we divide the sampled data into two groups, $S$ and $T$: $S$ contains the data for which we have determined the label ourselves (we explain below how to ensure that they are consistent with the best separator in the class) and $T$ contains the data for which we have explicitly requested a label. Now, somewhat counter-intuitively, the labels in $S$ are completely reliable, whereas the labels in $T$ could be inconsistent with the best separator. To decide if we are uncertain about the label of a new point $x$, we reduce to a supervised learning task: for each possible label $\hat{y} \in \{\pm 1\}$, we learn a hypothesis $h_{\hat{y}}$ consistent with the labels in $S \cup \{(x, \hat{y})\}$ and with minimal empirical error on $T$. If, say, the error of the hypothesis $h_{+1}$ is much larger than that of $h_{-1}$, we can safely infer that the best separator must also label $x$ with $-1$ without requesting a label; if the error difference is only modest, we explicitly request a label. Standard generalization bounds for an i.i.d. sample let us perform this test by comparing empirical errors on $S \cup T$.

The last claim may sound awfully suspicious, because $S \cup T$ is not i.i.d.! Indeed, this is in a sense the core sampling problem that has always plagued active learning: the labeled sample $T$ might not be i.i.d. (due to the filtering of examples based on an adaptive criterion), while $S$ only contains unlabeled examples (with made-up labels). Nevertheless, we prove that in our case, it is in fact correct to effectively pretend $S \cup T$ is an i.i.d. sample. A direct consequence is that the *label complexity* of our algorithm (the number of labels requested before achieving a desired error) is never much more than the usual sample complexity of supervised learning (and in some cases, is significantly less).

An important algorithmic detail is the specific choice of generalization bound we use in deciding whether to request a label or not. The usual additive bounds with rate $n^{-1/2}$ are too loose, e.g. we know in the zero-error case the rate should be $n^{-1}$. Our algorithm magnifies this small polynomial difference in the bound into an *exponential* difference in label complexity, so it is crucial for us to use a good bound. We use a normalized bound that takes into account the empirical error (computed on $S \cup T$) of the hypothesis in question.

In this paper, we present and analyze a simple agnostic active learning algorithm for general hypothesis classes of bounded VC dimension. It extends the selective sampling scheme of Cohn et al. [1] to the agnostic setting, using normalized generalization bounds, which we apply in a simple but subtle manner. For certain hypothesis classes and distributions, our analysis yields improved label complexity guarantees over the standard sample complexity of supervised learning. We also demonstrate such improvements experimentally.

## 1.1 Related work

Our algorithm extends the selective sampling scheme of Cohn et al. [1] (described above) to the agnostic setting. Most previous work on active learning either makes strong distributional assumptions (e.g. separability, uniform input distribution) [1–8], or is generally computationally prohibitive [2, 4, 9]. See [10] for a discussion of these results.

A natural way to formulate active learning in the *agnostic* setting is to ask the learner to return a hypothesis with error at most $\nu + \varepsilon$ (where $\nu$ is the error of the best hypothesis in

the specified class) using as few labels as possible. A basic constraint on the label complexity was pointed out by Kääriäinen [11], who showed that for any $\nu \in (0, 1/2)$, there are data distributions that force any active learner that achieves error at most $\nu + \varepsilon$ to request $\Omega((\nu/\varepsilon)^2)$ labels. The first rigorously-analyzed agnostic active learning algorithm, called $A^2$, was developed recently by Balcan, Beygelzimer, and Langford [9]. Like Cohn-Atlas-Ladner [1], this algorithm uses a region of uncertainty, although the lack of separability complicates matters and $A^2$ ends up explicitly maintaining an $\varepsilon$-net of the hypothesis space. Subsequently, Hanneke [12] characterized the label complexity of the $A^2$ algorithm in terms of a parameter called the *disagreement coefficient*.

Our work was inspired by both [1] and [9], and we have built heavily upon their insights. Our algorithm overcomes their complications by employing reductions to supervised learning.[1] We bound the label complexity of our method in terms of the same parameter as used for $A^2$ [12], and get a somewhat better dependence (linear rather than quadratic).

## 2 Preliminaries

### 2.1 Learning framework and uniform convergence

Let $\mathcal{X}$ be the input space, $\mathcal{D}$ a distribution over $\mathcal{X} \times \{\pm 1\}$ and $\mathcal{H}$ a class of hypotheses $h : \mathcal{X} \to \{\pm 1\}$ with VC dimension $\mathrm{vcdim}(\mathcal{H}) = d < \infty$ (the finiteness ensures the $n$th shatter coefficient $\mathcal{S}(\mathcal{H}, n)$ is at most $O(n^d)$ by Sauer's lemma). We denote by $\mathcal{D}_{\mathcal{X}}$ the marginal of $\mathcal{D}$ over $\mathcal{X}$. In our active learning model, the learner receives unlabeled data sampled from $\mathcal{D}_{\mathcal{X}}$; for any sampled point $x$, it can optionally request the label $y$ sampled from the conditional distribution at $x$. This process can be viewed as sampling $(x, y)$ from $\mathcal{D}$ and revealing only $x$ to the learner, keeping the label $y$ hidden unless the learner explicitly requests it. The error of a hypothesis $h$ under $\mathcal{D}$ is $\mathrm{err}_{\mathcal{D}}(h) = \Pr_{(x,y) \sim \mathcal{D}}[h(x) \neq y]$, and on a finite sample $Z \subset \mathcal{X} \times \{\pm 1\}$, the empirical error of $h$ is $\mathrm{err}(h, Z) = \sum_{(x,y) \in Z} \mathbb{1}[h(x) \neq y]/|Z|$, where $\mathbb{1}[\cdot]$ is the 0-1 indicator function. We assume for simplicity that the minimal error $\nu = \inf\{\mathrm{err}_{\mathcal{D}}(h) : h \in \mathcal{H}\}$ is achieved by a hypothesis $h^* \in \mathcal{H}$.

Our algorithm uses the following normalized uniform convergence bound [14, p. 200].

**Lemma 1** (Vapnik and Chervonenkis [15]). *Let $\mathcal{F}$ be a family of measurable functions $f : \mathcal{Z} \to \{0, 1\}$ over a space $\mathcal{Z}$. Denote by $\mathbb{E}_Z f$ the empirical average of $f$ over a subset $Z \subset \mathcal{Z}$. Let $\alpha_n = \sqrt{(4/n) \ln(8 \mathcal{S}(\mathcal{F}, 2n)/\delta)}$. If $Z$ is an i.i.d. sample of size $n$ from a fixed distribution over $\mathcal{Z}$, then, with probability at least $1 - \delta$, for all $f \in \mathcal{F}$:*

$$-\min\left(\alpha_n \sqrt{\mathbb{E}_Z f}, \alpha_n^2 + \alpha_n \sqrt{\mathbb{E} f}\right) \leq \mathbb{E} f - \mathbb{E}_Z f \leq \min\left(\alpha_n^2 + \alpha_n \sqrt{\mathbb{E}_Z f}, \alpha_n \sqrt{\mathbb{E} f}\right).$$

### 2.2 Disagreement coefficient

We will bound the label complexity of our algorithm in terms of (a slight variation of) the *disagreement coefficient* $\theta$ introduced in [12] for analyzing the label complexity of $A^2$.

**Definition 1.** *The* disagreement metric $\rho$ *on $\mathcal{H}$ is defined by $\rho(h, h') = \Pr_{x \sim \mathcal{D}_{\mathcal{X}}}[h(x) \neq h'(x)]$. The* disagreement coefficient $\theta = \theta(\mathcal{D}, \mathcal{H}, \varepsilon) > 0$ *is*

$$\theta = \sup\left\{ \frac{\Pr_{x \sim \mathcal{D}_{\mathcal{X}}}[\exists h \in B(h^*, r) \text{ s.t. } h(x) \neq h^*(x)]}{r} : r \geq \varepsilon + \nu \right\}$$

*where $B(h, r) = \{h' \in \mathcal{H} : \rho(h, h') < r\}$, $h^* = \arg\inf_{h \in \mathcal{H}} \mathrm{err}_{\mathcal{D}}(h)$, and $\nu = \mathrm{err}_{\mathcal{D}}(h^*)$.*

The quantity $\theta$ bounds the rate at which the disagreement mass of the ball $B(h^*, r)$ – the probability mass of points on which hypotheses in $B(h^*, r)$ disagree with $h^*$ – grows as a function of the radius $r$. Clearly, $\theta \leq 1/(\varepsilon + \nu)$; furthermore, it is a constant bounded

**Algorithm 1**
Input: stream $(x_1, x_2, \ldots, x_m)$ i.i.d. from $\mathcal{D}_{\mathcal{X}}$
Initially, $S_0 \leftarrow \emptyset$ and $T_0 \leftarrow \emptyset$.
For $n = 1, 2, \ldots, m$:

    1. For each $\hat{y} \in \{\pm 1\}$, let $h_{\hat{y}} \leftarrow \text{LEARN}_{\mathcal{H}}(S_{n-1} \cup \{(x_n, \hat{y})\}, T_{n-1})$.

    2. If $\text{err}(h_{-\hat{y}}, S_{n-1} \cup T_{n-1}) - \text{err}(h_{\hat{y}}, S_{n-1} \cup T_{n-1}) > \Delta_{n-1}$ for some $\hat{y} \in \{\pm 1\}$
       (or if no such $h_{-\hat{y}}$ is found)
       then $S_n \leftarrow S_{n-1} \cup \{(x_n, \hat{y})\}$ and $T_n \leftarrow T_{n-1}$.

    3. Else request $y_n$; $S_n \leftarrow S_{n-1}$ and $T_n \leftarrow T_{n-1} \cup \{(x_n, y_n)\}$.

Return $h_f = \text{LEARN}_{\mathcal{H}}(S_m, T_m)$.

Figure 1: The agnostic selective sampling algorithm. See (1) for how to set $\Delta_n$.

independently of $1/(\varepsilon + \nu)$ in several cases previously considered in the literature [12]. For example, if $\mathcal{H}$ is homogeneous linear separators and $\mathcal{D}_{\mathcal{X}}$ is the uniform distribution over the unit sphere in $\mathbb{R}^d$, then $\theta = \Theta(\sqrt{d})$.

## 3 Agnostic selective sampling

Here we state and analyze our general algorithm for agnostic active learning. The main techniques employed by the algorithm are reductions to a supervised learning task and generalization bounds applied to differences of empirical errors.

### 3.1 A general algorithm for agnostic active learning

Figure 1 states our algorithm in full generality. The input is a stream of $m$ unlabeled examples drawn i.i.d from $\mathcal{D}_{\mathcal{X}}$; for the time being, $m$ can be thought of as $\tilde{O}((d/\varepsilon)(1+\nu/\varepsilon))$ where $\varepsilon$ is the accuracy parameter.[2]

For $S, T \subset \mathcal{X} \times \{\pm 1\}$, let $\text{LEARN}_{\mathcal{H}}(S, T)$ denote a supervised learner that returns a hypothesis $h \in \mathcal{H}$ consistent with $S$, and with minimum error on $T$. Algorithm 1 maintains two sets of labeled examples, $S$ and $T$, each of which is initially empty. Upon receiving $x_n$, it learns two hypotheses, $h_{\hat{y}} = \text{LEARN}_{\mathcal{H}}(S \cup \{(x_n, \hat{y})\}, T)$ for $\hat{y} \in \{\pm 1\}$, and then compares their empirical errors on $S \cup T$. If the difference is large enough[3], it is possible to infer how $h^*$ labels $x_n$ (as we show in Lemma 3). In this case, the algorithm adds $x_n$, with this inferred label, to $S$. Otherwise, the algorithm requests the label $y_n$ and adds $(x_n, y_n)$ to $T$. Thus, $S$ contains examples with inferred labels consistent with $h^*$, and $T$ contains examples with their requested labels. Because $h^*$ might err on some examples in $T$, we just insist that $\text{LEARN}_{\mathcal{H}}$ find a hypothesis with minimal error on $T$. Meanwhile, by construction, $h^*$ is consistent with $S$, so we require $\text{LEARN}_{\mathcal{H}}$ to only consider hypotheses consistent with $S$.

### 3.2 Bounds for error differences

We still need to specify $\Delta_n$, the threshold value for error differences that determines whether the algorithm requests a label or not. Intuitively, $\Delta_n$ should reflect how closely empirical errors on a sample approximate true errors on the distribution $\mathcal{D}$.

The setting of $\Delta_n$ can only depend on observable quantities, so we first clarify the distinction between empirical errors on $S_n \cup T_n$ and those with respect to the true (hidden) labels.

**Definition 2.** *Let $S_n$ and $T_n$ be as defined in Algorithm 1. Let $S_n^!$ be the set of labeled examples identical to those in $S_n$, except with the true hidden labels swapped in. Thus, for example, $S_n^! \cup T_n$ is an i.i.d. sample from $\mathcal{D}$ of size $n$. Finally, let*

$$\text{err}_n^!(h) = \text{err}(h, S_n^! \cup T_n) \quad and \quad \text{err}_n(h) = \text{err}(h, S_n \cup T_n).$$

It is straightforward to apply Lemma 1 to empirical errors on $S_n^! \cup T_n$, i.e. to $\mathrm{err}_n^!(h)$, but we cannot use such bounds algorithmically: we do not request the true labels for points in $S_n$ and thus cannot reliably compute $\mathrm{err}_n^!(h)$. What we *can* compute are error differences $\mathrm{err}_n^!(h) - \mathrm{err}_n^!(h')$ for pairs of hypotheses $(h, h')$ that agree on (and make the same mistakes on) $S_n$, since for such pairs, we have $\mathrm{err}_n^!(h) - \mathrm{err}_n^!(h') = \mathrm{err}_n(h) - \mathrm{err}_n(h')$.

**Definition 3.** For a pair $(h, h') \in \mathcal{H} \times \mathcal{H}$, define $g_{h,h'}^+(x, y) = \mathbb{1}[h(x) \neq y \ \wedge \ h'(x) = y]$ and $g_{h,h'}^-(x, y) = \mathbb{1}[h(x) = y \ \wedge \ h'(x) \neq y]$.

With this notation, we have $\mathrm{err}(h, Z) - \mathrm{err}(h', Z) = \mathbb{E}_Z[g_{h,h'}^+] - \mathbb{E}_Z[g_{h,h'}^-]$ for any $Z \subset \mathcal{X} \times \{\pm 1\}$. Now, applying Lemma 1 to $\mathcal{G} = \{g_{h,h'}^+ : (h, h') \in \mathcal{H} \times \mathcal{H}\} = \{g_{h,h'}^- : (h, h') \in \mathcal{H} \times \mathcal{H}\}$, and noting that $\mathcal{S}(\mathcal{G}, n) \leq \mathcal{S}(\mathcal{H}, n)^2$, gives the following lemma.

**Lemma 2.** Let $\alpha_n = \sqrt{(4/n) \ln(8\mathcal{S}(\mathcal{H}, 2n)^2/\delta)}$. With probability at least $1 - \delta$ over an i.i.d. sample $Z$ of size $n$ from $\mathcal{D}$, we have for all $(h, h') \in \mathcal{H} \times \mathcal{H}$,

$$\mathrm{err}(h, Z) - \mathrm{err}(h', Z) \leq \mathrm{err}_{\mathcal{D}}(h) - \mathrm{err}_{\mathcal{D}}(h') + \alpha_n^2 + \alpha_n \left( \sqrt{\mathbb{E}_Z[g_{h,h'}^+]} + \sqrt{\mathbb{E}_Z[g_{h,h'}^-]} \right).$$

**Corollary 1.** Let $\beta_n = \sqrt{(4/n) \ln(8(n^2 + n)\mathcal{S}(\mathcal{H}, 2n)^2/\delta)}$. Then, with probability at least $1 - \delta$, for all $n \geq 1$ and all $(h, h') \in \mathcal{H} \times \mathcal{H}$ consistent with $S_n$, we have

$$\mathrm{err}_n(h) - \mathrm{err}_n(h') \leq \mathrm{err}_{\mathcal{D}}(h) - \mathrm{err}_{\mathcal{D}}(h') + \beta_n^2 + \beta_n(\sqrt{\mathrm{err}_n(h)} + \sqrt{\mathrm{err}_n(h')}).$$

*Proof.* Applying Lemma 2 to each $S_n^! \cup T_n$ (replacing $\delta$ with $\delta/(n^2 + n)$) and a union bound implies, with probability at least $1 - \delta$, the bounds in Lemma 2 hold simultaneously for all $n \geq 1$ and all $(h, h') \in \mathcal{H}^2$ with $S_n^! \cup T_n$ in place of $Z$. The corollary follows because $\mathrm{err}_n^!(h) - \mathrm{err}_n^!(h') = \mathrm{err}_n(h) - \mathrm{err}_n(h')$; and because $g_{h,h'}^+(x, y) \leq \mathbb{1}[h(x) \neq y]$ and $g_{h,h'}^-(x, y) \leq \mathbb{1}[h'(x) \neq y]$ for $(h, h')$ consistent with $S_n$, so $\mathbb{E}_{S_n^! \cup T_n}[g_{h,h'}^+] \leq \mathrm{err}_n(h)$ and $\mathbb{E}_{S_n^! \cup T_n}[g_{h,h'}^-] \leq \mathrm{err}_n(h')$. $\square$

Corollary 1 implies that we can effectively apply the normalized uniform convergence bounds from Lemma 1 to empirical error differences on $S_n \cup T_n$, *even though $S_n \cup T_n$ is not an i.i.d. sample from $\mathcal{D}$*. In light of this, we use the following setting of $\Delta_n$:

$$\Delta_n := \beta_n^2 + \beta_n \left( \sqrt{\mathrm{err}_n(h_{+1})} + \sqrt{\mathrm{err}_n(h_{-1})} \right) \tag{1}$$

where $\beta_n = \sqrt{(4/n) \ln(8(n^2 + n)\mathcal{S}(\mathcal{H}, 2n)^2/\delta)} = \tilde{O}(\sqrt{d \log n/n})$ as per Corollary 1.

### 3.3 Correctness and fall-back analysis

We now justify our setting of $\Delta_n$ with a correctness proof and fall-back guarantee.

**Lemma 3.** With probability at least $1 - \delta$, the hypothesis $h^* = \arg \inf_{h \in \mathcal{H}} \mathrm{err}_{\mathcal{D}}(h)$ is consistent with $S_n$ for all $n \geq 0$ in Algorithm 1.

*Proof.* Apply the bounds in Corollary 1 and proceed by induction on $n$. The base case is trivial since $S_0 = \emptyset$. Now assume $h^*$ is consistent with $S_n$. Suppose upon receiving $x_{n+1}$, we discover $\mathrm{err}_n(h_{+1}) - \mathrm{err}_n(h_{-1}) > \Delta_n$. We will show that $h^*(x_{n+1}) = -1$ (assume both $h_{+1}$ and $h_{-1}$ exist, since it is clear $h^*(x_{n+1}) = -1$ if $h_{+1}$ does not exist). Suppose for the sake of contradiction that $h^*(x_{n+1}) = +1$. We know the that $\mathrm{err}_n(h^*) \geq \mathrm{err}_n(h_{+1})$ (by inductive hypothesis) and $\mathrm{err}_n(h_{+1}) - \mathrm{err}_n(h_{-1}) > \beta_n^2 + \beta_n(\sqrt{\mathrm{err}_n(h_{+1})} + \sqrt{\mathrm{err}_n(h_{-1})})$. In particular, $\mathrm{err}_n(h_{+1}) > \beta_n^2$. Therefore,

$$\mathrm{err}_n(h^*) - \mathrm{err}_n(h_{-1}) = (\mathrm{err}_n(h^*) - \mathrm{err}_n(h_{+1})) + (\mathrm{err}_n(h_{+1}) - \mathrm{err}_n(h_{-1}))$$
$$> \sqrt{\mathrm{err}_n(h_{+1})}(\sqrt{\mathrm{err}_n(h^*)} - \sqrt{\mathrm{err}_n(h_{+1})}) + \beta_n^2 + \beta_n(\sqrt{\mathrm{err}_n(h_{+1})} + \sqrt{\mathrm{err}_n(h_{-1})})$$
$$> \beta_n(\sqrt{\mathrm{err}_n(h^*)} - \sqrt{\mathrm{err}_n(h_{+1})}) + \beta_n^2 + \beta_n(\sqrt{\mathrm{err}_n(h_{+1})} + \sqrt{\mathrm{err}_n(h_{-1})})$$
$$= \beta_n^2 + \beta_n(\sqrt{\mathrm{err}_n(h^*)} + \sqrt{\mathrm{err}_n(h_{-1})}).$$

Now Corollary 1 implies that $\mathrm{err}_{\mathcal{D}}(h^*) > \mathrm{err}_{\mathcal{D}}(h_{-1})$, a contradiction. $\square$

**Theorem 1.** *Let $\nu = \inf_{h \in \mathcal{H}} \text{err}_\mathcal{D}(h)$ and $d = \text{vcdim}(\mathcal{H})$. There exists a constant $c > 0$ such that the following holds. If Algorithm 1 is given a stream of $m$ unlabeled examples, then with probability at least $1 - \delta$, the algorithm returns a hypothesis with error at most $\nu + c \cdot ((1/m)(d \log m + \log(1/\delta)) + \sqrt{(\nu/m)(d \log m + \log(1/\delta))})$.*

*Proof.* Lemma 3 implies that $h^*$ is consistent with $S_m$ with probability at least $1 - \delta$. Using the same bounds from Corollary 1 (already applied in Lemma 3) on $h^*$ and $h_f$ together with the fact $\text{err}_m(h_f) \leq \text{err}_m(h^*)$, we have $\text{err}_\mathcal{D}(h_f) \leq \nu + \beta_m^2 + \beta_m\sqrt{\nu} + \beta_m\sqrt{\text{err}_\mathcal{D}(h_f)}$, which in turn implies $\text{err}_\mathcal{D}(h_f) \leq \nu + 3\beta_m^2 + 2\beta_m\sqrt{\nu}$. $\square$

So, Algorithm 1 returns a hypothesis with error at most $\nu + \varepsilon$ when $m = \tilde{O}((d/\varepsilon)(1 + \nu/\varepsilon))$; this is (asymptotically) the usual sample complexity of supervised learning. Since the algorithm requests at most $m$ labels, its label complexity is always at most $\tilde{O}((d/\varepsilon)(1 + \nu/\varepsilon))$.

### 3.4 Label complexity analysis

We can also bound the label complexity of our algorithm in terms of the disagreement coefficient $\theta$. This yields tighter bounds when $\theta$ is bounded independently of $1/(\varepsilon + \nu)$. The key to deriving our label complexity bounds based on $\theta$ is noting that the probability of requesting the $(n + 1)$th label is intimately related to $\theta$ and $\Delta_n$ (see [10] for the complete proof).

**Lemma 4.** *There exists a constant $c > 0$ such that, with probability at least $1 - 2\delta$, for all $n \geq 1$, the following holds. Let $h^*(x_{n+1}) = \hat{y}$ where $h^* = \arg\inf_{h \in \mathcal{H}} \text{err}_\mathcal{D}(h)$. Then, the probability that Algorithm 1 requests the label $y_{n+1}$ is $\Pr_{x_{n+1} \sim \mathcal{D}_\mathcal{X}}[\text{Request } y_{n+1}] \leq c \cdot \theta \cdot (\nu + \beta_n^2)$, where $\theta = \theta(\mathcal{D}, \mathcal{H}, 3\beta_m^2 + 2\beta_m\sqrt{\nu})$ is the disagreement coefficient, $\nu = \text{err}_\mathcal{D}(h^*)$, and $\beta_n = \tilde{O}(\sqrt{d \log n / n})$ is as defined in Corollary 1.*

Now we give our main label complexity bound for agnostic active learning.

**Theorem 2.** *Let $m$ be the number of unlabeled data given to Algorithm 1, $d = \text{vcdim}(\mathcal{H})$, $\nu = \inf_{h \in \mathcal{H}} \text{err}_\mathcal{D}(h)$, $\beta_m$ as defined in Corollary 1, and $\theta = \theta(\mathcal{D}, \mathcal{H}, 3\beta_m^2 + 2\beta_m\sqrt{\nu})$. There exists a constant $c_1 > 0$ such that for any $c_2 \geq 1$, with probability at least $1 - 2\delta$:*

1. *If $\nu \leq (c_2 - 1)\beta_m^2$, Algorithm 1 returns a hypothesis with error as bounded in Theorem 1 and the expected number of labels requested is at most*

$$1 + c_1 c_2 \theta \cdot \left( d \log^2 m + \log \frac{1}{\delta} \log m \right).$$

2. *Else, the same holds except the expected number of labels requested is at most*

$$1 + c_1 \theta \cdot \left( \nu m + d \log^2 m + \log \frac{1}{\delta} \log m \right).$$

*Furthermore, if $L$ is the expected number of labels requested as per above, then with probability at least $1 - \delta'$, the algorithm requests no more than $L + \sqrt{3L \log(1/\delta')}$ labels.*

*Proof.* Follows from Lemma 4 and a Chernoff bound for the Poisson trials $\mathbb{1}[\text{Request } y_n]$. $\square$

With the substitution $\varepsilon = 3\beta_m^2 + 2\beta_m\sqrt{\nu}$ as per Theorem 1, Theorem 2 entails that for any hypothesis class and data distribution for which the disagreement coefficient $\theta = \theta(\mathcal{D}, \mathcal{H}, \varepsilon)$ is bounded independently of $1/(\varepsilon + \nu)$ (see [12] for some examples), Algorithm 1 only needs $\tilde{O}(\theta d \log^2(1/\varepsilon))$ labels to achieve error $\varepsilon \approx \nu$ and $\tilde{O}(\theta d(\log^2(1/\varepsilon) + (\nu/\varepsilon)^2))$ labels to achieve error $\varepsilon \ll \nu$. The latter matches the dependence on $\nu/\varepsilon$ in the $\Omega((\nu/\varepsilon)^2)$ lower bound [11].

The linear dependence on $\theta$ improves on the quadratic dependence required by $A^2$ [12][4]. For an illustrative consequence of this, suppose $\mathcal{D}_\mathcal{X}$ is the uniform distribution on the sphere

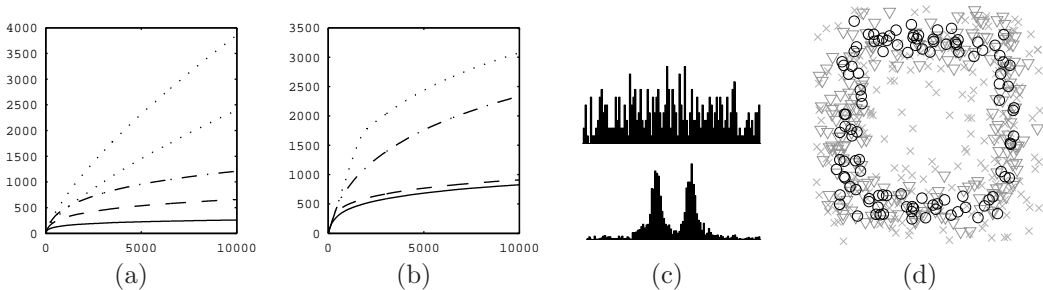

Figure 2: (a & b) Labeling rate plots. The plots show the number of labels requested (vertical axis) versus the total number of points seen (labeled + unlabeled, horizontal axis) using Algorithm 1. (a) $\mathcal{H}$ = thresholds: under random misclassification noise with $\nu = 0$ (solid), 0.1 (dashed), 0.2 (dot-dashed); under the boundary noise model with $\nu = 0.1$ (lower dotted), 0.2 (upper dotted). (b) $\mathcal{H}$ = intervals: under random misclassification with $(p_+, \nu) = (0.2, 0.0)$ (solid), $(0.1, 0.0)$ (dashed), $(0.2, 0.1)$ (dot-dashed), $(0.1, 0.1)$ (dotted). (c & d) Locations of label requests. (c) $\mathcal{H}$ = intervals, $h^* = [0.4, 0.6]$. The top histogram shows the locations of first 400 label requests (the x-axis is the unit interval); the bottom histogram is for all (2141) label requests. (d) $\mathcal{H}$ = boxes, $h^* = [0.15, 0.85]^2$. The first 200 requests occurred at the $\times$s, the next 200 at the $\bigtriangledown$s, and the final 109 at the $\bigcirc$s.

in $\mathbb{R}^d$ and $\mathcal{H}$ is homogeneous linear separators; in this case, $\theta = \Theta(\sqrt{d})$. Then the label complexity of $A^2$ depends at least quadratically on the dimension, whereas the corresponding dependence for our algorithm is $d^{3/2}$.

## 4  Experiments

We implemented Algorithm 1 in a few simple cases to experimentally demonstrate the label complexity improvements. In each case, the data distribution $\mathcal{D}_\mathcal{X}$ was uniform over $[0, 1]$; the stream length was $m = 10000$, and each experiment was repeated 20 times with different random seeds. Our first experiment studied linear thresholds on the line. The target hypothesis was fixed to be $h^*(x) = \text{sign}(x - 0.5)$. For this hypothesis class, we used two different noise models, each of which ensured $\inf_{h \in \mathcal{H}} \text{err}_\mathcal{D}(h) = \text{err}_\mathcal{D}(h^*) = \nu$ for a pre-specified $\nu \in [0, 1]$. The first model was random misclassification: for each point $x \sim \mathcal{D}_\mathcal{X}$, we independently labeled it $h^*(x)$ with probability $1 - \nu$ and $-h^*(x)$ with probability $\nu$. In the second model (also used in [7]), for each point $x \sim \mathcal{D}_\mathcal{X}$, we independently labeled it $+1$ with probability $(x - 0.5)/(4\nu) + 0.5$ and $-1$ otherwise, thus concentrating the noise near the boundary. Our second experiment studied intervals on the line. Here, we only used random misclassification, but we varied the target interval length $p_+ = \text{Pr}_{x \sim \mathcal{D}_\mathcal{X}}[h^*(x) = +1]$.

The results show that the number of labels requested by Algorithm 1 was exponentially smaller than the total number of data seen ($m$) under the first noise model, and was polynomially smaller under the second noise model (see Figure 2 (a & b); we verified the polynomial vs. exponential distinction on separate log-log scale plots). In the case of intervals, we observe an initial phase (of duration roughly $\propto 1/p_+$) in which every label is requested, followed by a more efficient phase, confirming the known active-learnability of this class [4,12]. These improvements show that our algorithm needed significantly fewer labels to achieve the same error as a standard supervised algorithm that uses labels for all points seen.

As a sanity check, we examined the locations of data for which Algorithm 1 requested a label. We looked at two particular runs of the algorithm: the first was with $\mathcal{H}$ = intervals, $p_+ = 0.2$, $m = 10000$, and $\nu = 0.1$; the second was with $\mathcal{H}$ = boxes ($d = 2$), $p_+ = 0.49$, $m = 1000$, and $\nu = 0.01$. In each case, the data distribution was uniform over $[0, 1]^d$, and the noise model was random misclassification. Figure 2 (c & d) shows that, early on, labels were requested everywhere. But as the algorithm progressed, label requests concentrated near the boundary of the target hypothesis.

# 5 Conclusion and future work

We have presented a simple and natural approach to agnostic active learning. Our extension of the selective sampling scheme of Cohn, et al. [1]

1. simplifies the maintenance of the region of uncertainty with a reduction to supervised learning, and
2. guards against noise with a subtle algorithmic application of generalization bounds.

Our algorithm relies on a threshold parameter $\Delta_n$ for comparing empirical errors. We prescribe a very simple and natural choice for $\Delta_n$ – a normalized generalization bound from supervised learning – but one could hope for a more clever or aggressive choice, akin to those in [6] for linear separators.

Finding consistent hypotheses when data is separable is often a simple task. In such cases, reduction-based active learning algorithms can be relatively efficient (answering some questions posed in [16]). On the other hand, agnostic learning suffers from severe computational intractability for many hypothesis classes (e.g. [17]), and of course, agnostic active learning is at least as hard in the worst case. Our reduction is relatively benign in that the learning problems created are only over samples from the original distribution, so we do not create pathologically hard instances (like those arising from hardness reductions) unless they are inherent in the data. Nevertheless, an important research direction is to develop algorithms that only require solving tractable (e.g. convex) optimization problems. A similar reduction-based scheme may be possible.

## Footnotes

[1]It has been noted that the Cohn-Atlas-Ladner scheme can easily be made tractable using a reduction to supervised learning in the separable case [13, p. 68]. Although our algorithm is most naturally seen as an extension of Cohn-Atlas-Ladner, a similar reduction to supervised learning (in the agnostic setting) can be used for $A^2$ [10].

[2]The $\tilde{O}$ notation suppresses $\log 1/\delta$ and terms polylogarithmic in those that appear.

[3]If $\text{LEARN}_{\mathcal{H}}$ cannot find a hypothesis consistent with $S \cup \{(x_n, y)\}$ for some $y$, then it is clear that $h^*(x) = -y$. In this case, we simply add $(x_n, -y)$ to $S$, regardless of $\Delta_{n-1}$.

[4]It may be possible to reduce $A^2$'s quadratic dependence to a linear dependence by using normalized bounds, as we do here.

## References

[1] D. Cohn, L. Atlas, and R. Ladner. Improving generalization with active learning. *Machine Learning*, 15(2):201–221, 1994.

[2] Y. Freund, H. Seung, E. Shamir, and N. Tishby. Selective sampling using the query by committee algorithm. *Machine Learning*, 28(2):133–168, 1997.

[3] S. Dasgupta, A. Kalai, and C. Monteleoni. Analysis of perceptron-based active learning. In *COLT*, 2005.

[4] S. Dasgupta. Coarse sample complexity bounds for active learning. In *NIPS*, 2005.

[5] S. Hanneke. Teaching dimension and the complexity of active learning. In *COLT*, 2007.

[6] M.-F. Balcan, A. Broder, and T. Zhang. Margin based active learning. In *COLT*, 2007.

[7] R. Castro and R. Nowak. Upper and lower bounds for active learning. In *Allerton Conference on Communication, Control and Computing*, 2006.

[8] R. Castro and R. Nowak. Minimax bounds for active learning. In *COLT*, 2007.

[9] M.-F. Balcan, A. Beygelzimer, and J. Langford. Agnostic active learning. In *ICML*, 2006.

[10] S. Dasgupta, D. Hsu, and C. Monteleoni. A general agnostic active learning algorithm. *UCSD Technical Report CS2007-0898*, `http://www.cse.ucsd.edu/~djhsu/papers/cal.pdf`, 2007.

[11] M. Kääriäinen. Active learning in the non-realizable case. In *ALT*, 2006.

[12] S. Hanneke. A bound on the label complexity of agnostic active learning. In *ICML*, 2007.

[13] C. Monteleoni. *Learning with online constraints: shifting concepts and active learning*. PhD Thesis, MIT Computer Science and Artificial Intelligence Laboratory, 2006.

[14] O. Bousquet, S. Boucheron, and G. Lugosi. Introduction to statistical learning theory. *Lecture Notes in Artificial Intelligence*, 3176:169–207, 2004.

[15] V. Vapnik and A. Chervonenkis. On the uniform convergence of relative frequencies of events to their probabilities. *Theory of Probability and its Applications*, 16:264–280, 1971.

[16] C. Monteleoni. Efficient algorithms for general active learning. In *COLT*. Open problem, 2006.

[17] V. Guruswami and P. Raghavendra. Hardness of learning halfspaces with noise. In *FOCS*, 2006.

